# CRICKET WIND DETECTION

John P. Miller

*Neurobiology Group, University of California,*
*Berkeley, California 94720, U.S.A.*

A great deal of interest has recently been focused on theories concerning parallel distributed processing in central nervous systems. In particular, many researchers have become very interested in the structure and function of "computational maps" in sensory systems. As defined in a recent review (Knudsen et al, 1987), a "map" is an array of nerve cells, within which there is a systematic variation in the "tuning" of neighboring cells for a particular parameter. For example, the projection from retina to visual cortex is a relatively simple topographic map; each cortical hypercolumn itself contains a more complex "computational" map of preferred line orientation representing the angle of tilt of a simple line stimulus.

The overall goal of the research in my lab is to determine how a relatively complex mapped sensory system extracts and encodes information from external stimuli. The preparation we study is the cercal sensory system of the cricket, *Acheta domesticus.* Crickets (and many other insects) have two antenna-like appendages at the rear of their abdomen, covered with hundreds of "filiform" hairs, resembling bristles on a bottle brush. Deflection of these filiform hairs by wind currents activates mechanosensory receptors, which project into the terminal abdominal ganglion to form a topographic representation (or "map") of "wind space". Primary sensory interneurons having

dendritic branches within this afferent map of wind space are selectively activated by wind stimuli with "relevant" parameters, and generate action potentials at frequencies that depend upon the value of those parameters. The "relevant" parameters are thought to be the direction, velocity, and acceleration of wind currents directed at the animal (Shimozawa & Kanou, 1984a & b). There are only ten pairs of these interneurons which carry the system's output to higher centers. All ten of these output units are identified, and all can be monitored individually with intracellular electrodes or simultaneously with extracellular electrodes. The following specific questions are currently being addressed: What are the response properties of the sensory receptors, and what are the I/O properties of the receptor layer as a whole? What are the response properties of all the units in the output layer? Is all of the direction, velocity and acceleration information that is extracted at the receptor layer also available at the output layer? How is that information encoded? Are any higher order "features" also encoded? What is the overall threshold, sensitivity and dynamic range of the system as a whole for detecting features of wind stimuli?

Michael Landolfa is studying the sensory neurons which serve as the inputs to the cercal system. The sensory cell layer consists of about 1000 afferent neurons, each of which innervates a single mechanosensory hair on the cerci. The input/output relationships of single sensory neurons were characterized by recording from an afferent axon while presenting appropriate stimuli to the sensory hairs. The primary results were as follows: 1) Afferents are directionally sensitive. Graphs of afferent response amplitude versus wind direction are approximately sinusoidal, with distinct preferred and anti-preferred directions. 2) Afferents are velocity sensitive. Each afferent encodes wind

velocity over a range of approximately 1.5 log units. 3) Different afferents have different velocity thresholds. The overlap of these different sensitivity curves insures that the system as a whole can encode wind velocities that span several log units. 4) The nature of the afferent response to deflection of its sensory hair indicates that the parameter transduced by the afferent is not hair displacement, but change in hair displacement. Thus, a significant portion of the processing which occurs within the cercal sensory system is accomplished at the level of the sensory afferents.

This information about the direction and velocity of wind stimuli is encoded by the relative firing rates of at least 10 pairs of identified sensory interneurons. A full analysis of the input/output properties of this system requires that the activity of these output neurons be monitored simultaneously. Shai Gozani has implemented a computer-based system capable of extracting the firing patterns of individual neurons from multi-unit recordings. For these experiments, extracellular electrodes were arrayed along the abdominal nerve cord in each preparation. Wind stimuli of varying directions, velocities and frequencies were presented to the animals. The responses of the cells were analyzed by spike descrimination software based on an algorithm originally developed by Roberts and Hartline (1975). The algorithm employs multiple linear filters, and is capable of descriminating spikes that were coincident in time. The number of spikes that could be descriminated was roughly equal to the number of independent electrodes. These programs are very powerful, and may be of much more general utility for researchers working on other invertebrate and vertebrate preparations. Using these programs and protocols, we have characterized the output of the cercal sensory system in terms of the simultaneous activity patterns of several pairs of identified

interneurons.

The results of these multi-unit recording studies, as well as studies using single intracellular electrodes, have yielded information about the directional tuning and velocity sensitivity of the first order sensory interneurons. Tuning curves representing interneuron response amplitude versus wind direction are approximately sinusoidal, as was the case for the sensory afferents. Sensitivity curves representing interneuron response amplitude versus wind velocity are sigmoidal, with "operating ranges" of about 1.5 log units. The interneurons are segregated into several distinct classes having different but overlapping operating ranges, such that the direction and velocity of any wind stimulus can be uniquely represented as the ratio of activity in the different interneurons. Thus, the overlap of the different direction and velocity sensitivity curves in approximately 20 interneurons insures that the system as a whole can encode the characteristics of wind stimuli having directions that span 360 degrees and velocities that span at least 4 orders of magnitude.

We are particularly interested in the mechanisms underlying directional sensitivity in some of the first-order sensory interneurons. Identified interneurons with different morphologies have very different directional sensitivities. The excitatory receptive fields of the different interneurons have been shown to be directly related to the position of their dendrites within the topographic map of wind space formed by the filiform afferents discussed above (Bacon & Murphey, 1984; Jacobs & Miller,1985; Jacobs, Miller & Murphey, 1986). The precise shapes of the directional tuning curves have been shown to be dependent upon two additional factors. First, local inhibitory interneurons can have a stong influence over a cell's response by shunting excitatory inputs from particular directions, and by reducing spontaneous activity during

stimuli from a cells "null" direction. Second, the "electroanatomy" of a neuron's dendritic branches determines the relative weighting of synaptic inputs onto its different arborizations.

Some specific aims of our continuing research are as follows: 1) to characterize the distribution of all synaptic inputs onto several different types of identified interneurons, 2) to measure the functional properties of individual dendrites of these cell types, 3) to locate the spike initiating zones of the cells, and 4) to synthesize a quantitative explanation of signal processing by each cell. Steps 1, 2 & 3 are being accomplished through electrophysiological experiments. Step 4 is being accomplished by developing a compartmental model for each cell type and testing the model through further physiological experiments. These computer modeling studies are being carried out by Rocky Nevin and John Tromp. For these models, the structure of each interneuron's dendritic branches are of particular functional importance, since the flow of bioelectrical currents through these branches determine how signals received from "input" cells are "integrated" and transformed into meaningful output which is transmitted to higher centers.

We are now at a point where we can begin to understand the operation of the system as a whole in terms of the structure, function and synaptic connectivity of the individual neurons. The proposed studies will also lay the technical and theoretical groundwork for future studies into the nature of signal "decoding" and higher-order processing in this preparation, mechanisms underlying the development, self-organization and regulative plasticity of units within this computational map, and perhaps information processing in more complex mapped sensory systems.

# REFERENCES

Bacon, J.P. and Murphey, R.K. (1984) Receptive fields of cricket (*Acheta domesticus*) are determined by their dendritic structure. *J.Physiol.(Lond)* 352:601

Jacobs, G.A. and Miller, J.P. (1985) Functional properties of individual neuronal branches isolated *in situ* by laser photoinactivation. *Science*, 228: 344-346

Jacobs, G.A., Miller, J.P. and Murphey, R.K. (1986) Cellular mechanisms underlying directional sensitivity of an identified sensory interneuron. *J. Neurosci.* 6(8): 2298-2311

Knudsen, E.I., S. duLac and Esterly, S.D. (1987) Computational maps in the brain. *Annual Review of Neuroscience* 10: 41-66

Roberts, W.M. and Hartline, D.K. (1975) Separation of multi-unit nerve impulse trains by a multi-channel linear filter algorithm. *Brain Res.* 94: 141- 149.

Shimozawa, T. and Kanou, M. (1984a) Varieties of filiform hairs: range fractionation by sensory afferents and cercal interneurons of a cricket. *J. Comp. Physiol. A.* 155: 485-493

Shimozawa, T. and Kanou, M. (1984b) The aerodynamics and sensory physiology of range fractionation in the cercal filiform sensilla of the cricket Gryllus bimaculatus. *J. Comp. Physiol. A.* 155: 495-505

